# Self-Tuning Spectral Clustering

**Lihi Zelnik-Manor**
Department of Electrical Engineering
California Institute of Technology
Pasadena, CA 91125, USA
lihi@vision.caltech.edu

**Pietro Perona**
Department of Electrical Engineering
California Institute of Technology
Pasadena, CA 91125, USA
perona@vision.caltech.edu

http://www.vision.caltech.edu/lihi/Demos/SelfTuningClustering.html

## Abstract

We study a number of open issues in spectral clustering: (i) Selecting the appropriate scale of analysis, (ii) Handling multi-scale data, (iii) Clustering with irregular background clutter, and, (iv) Finding automatically the number of groups. We first propose that a 'local' scale should be used to compute the affinity between each pair of points. This local scaling leads to better clustering especially when the data includes multiple scales and when the clusters are placed within a cluttered background. We further suggest exploiting the structure of the eigenvectors to infer automatically the number of groups. This leads to a new algorithm in which the final randomly initialized k-means stage is eliminated.

## 1 Introduction

Clustering is one of the building blocks of modern data analysis. Two commonly used methods are K-means and learning a mixture-model using EM. These methods, which are based on estimating explicit models of the data, provide high quality results when the data is organized according to the assumed models. However, when it is arranged in more complex and unknown shapes, these methods tend to fail. An alternative clustering approach, which was shown to handle such structured data is spectral clustering. It does not require estimating an explicit model of data distribution, rather a spectral analysis of the matrix of point-to-point similarities. A first set of papers suggested the method based on a set of heuristics (e.g., [8, 9]). A second generation provided a level of theoretical analysis, and suggested improved algorithms (e.g., [6, 10, 5, 4, 3]).

There are still open issues: (i) Selection of the appropriate scale in which the data is to be analyzed, (ii) Clustering data that is distributed according to different scales, (iii) Clustering with irregular background clutter, and, (iv) Estimating automatically the number of groups. We show here that it is possible to address these issues and propose ideas to tune the parameters automatically according to the data.

### 1.1 Notation and the Ng-Jordan-Weiss (NJW) Algorithm

The analysis and approaches suggested in this paper build on observations presented in [5]. For completeness of the text we first briefly review their algorithm.

Given a set of $n$ points $S = \{s_1, \ldots, s_n\}$ in $\mathcal{R}^l$ cluster them into $C$ clusters as follows:

1. Form the affinity matrix $A \in \mathcal{R}^{n \times n}$ defined by $A_{ij} = \exp\left(\frac{-d^2(s_i, s_j)}{\sigma^2}\right)$ for $i \neq j$ and $A_{ii} = 0$, where $d(s_i, s_j)$ is some distance function, often just the Euclidean

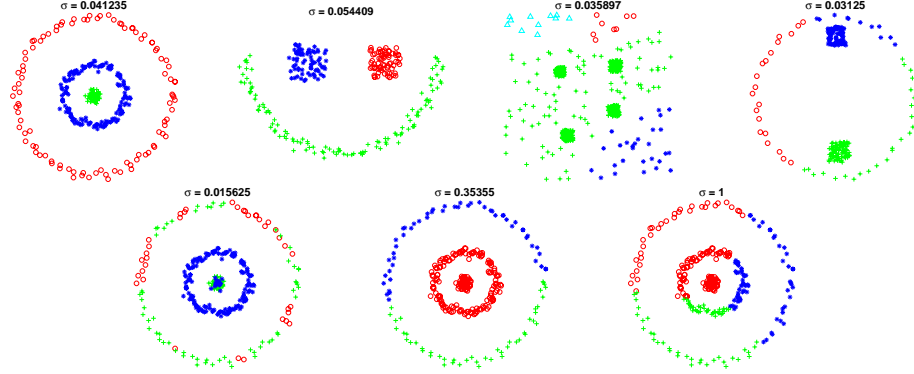

Figure 1: **Spectral clustering without local scaling** (using the NJW algorithm.) *Top row: When the data incorporates multiple scales standard spectral clustering fails. Note, that the optimal $\sigma$ for each example (displayed on each figure) turned out to be different. Bottom row: Clustering results for the top-left point-set with different values of $\sigma$. This highlights the high impact $\sigma$ has on the clustering quality. In all the examples, the number of groups was set manually. The data points were normalized to occupy the $[-1, 1]^2$ space.*

      distance between the vectors $s_i$ and $s_j$. $\sigma$ is a scale parameter which is further discussed in Section 2.

2. Define $D$ to be a diagonal matrix with $D_{ii} = \sum_{j=1}^{n} A_{ij}$ and construct the normalized affinity matrix $L = D^{-1/2}AD^{-1/2}$.

3. Manually select a desired number of groups $C$.

4. Find $x_1, \ldots, x_C$, the $C$ largest eigenvectors of $L$, and form the matrix $X = [x_1, \ldots, x_C] \in \mathcal{R}^{n \times C}$.

5. Re-normalize the rows of $X$ to have unit length yielding $Y \in \mathcal{R}^{n \times C}$, such that $Y_{ij} = X_{ij}/(\sum_j X_{ij}^2)^{1/2}$.

6. Treat each row of $Y$ as a point in $\mathcal{R}^C$ and cluster via k-means.

7. Assign the original point $s_i$ to cluster $c$ if and only if the corresponding row $i$ of the matrix $Y$ was assigned to cluster $c$.

In Section 2 we analyze the effect of $\sigma$ on the clustering and suggest a method for setting it automatically. We show that this allows handling multi-scale data and background clutter. In Section 3 we suggest a scheme for finding automatically the number of groups $C$. Our new spectral clustering algorithm is summarized in Section 4. We conclude with a discussion in Section 5.

## 2   Local Scaling

As was suggested by [6] the scaling parameter is some measure of when two points are considered similar. This provides an intuitive way for selecting possible values for $\sigma$. The selection of $\sigma$ is commonly done manually. Ng et al. [5] suggested selecting $\sigma$ automatically by running their clustering algorithm repeatedly for a number of values of $\sigma$ and selecting the one which provides least distorted clusters of the rows of $Y$. This increases significantly the computation time. Additionally, the range of values to be tested still has to be set manually. Moreover, when the input data includes clusters with different local statistics there may not be a singe value of $\sigma$ that works well for all the data. Figure 1 illustrates the high impact $\sigma$ has on clustering. When the data contains multiple scales, even using the optimal $\sigma$ fails to provide good clustering (see examples at the right of top row).

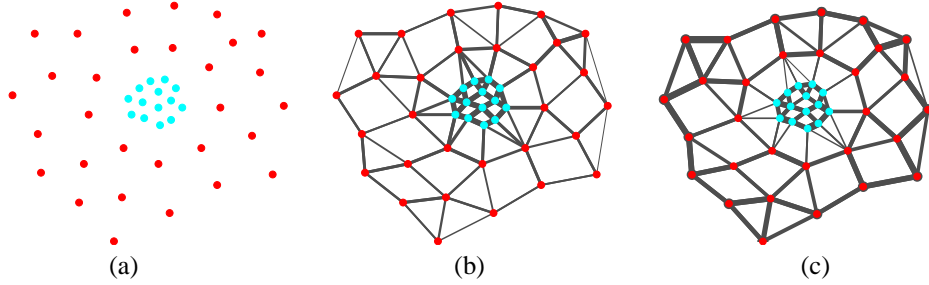

<center>(a)            (b)            (c)</center>

Figure 2: **The effect of local scaling.** *(a) Input data points. A tight cluster resides within a background cluster. (b) The affinity between each point and its surrounding neighbors is indicated by the thickness of the line connecting them. The affinities across clusters are larger than the affinities within the background cluster. (c) The corresponding visualization of affinities after local scaling. The affinities across clusters are now significantly lower than the affinities within any single cluster.*

**Introducing Local Scaling:** Instead of selecting a single scaling parameter $\sigma$ we propose to calculate a local scaling parameter $\sigma_i$ for each data point $s_i$. The distance from $s_i$ to $s_j$ as 'seen' by $s_i$ is $d(s_i, s_j)/\sigma_i$ while the converse is $d(s_j, s_i)/\sigma_j$. Therefore the square distance $d^2$ of the earlier papers may be generalized as $d(s_i, s_j)d(s_j, s_i)/\sigma_i\sigma_j = d^2(s_i, s_j)/\sigma_i\sigma_j$ The affinity between a pair of points can thus be written as:

$$\hat{A}_{ij} = \exp\left(\frac{-d^2(s_i, s_j)}{\sigma_i\sigma_j}\right) \tag{1}$$

Using a specific scaling parameter for each point allows self-tuning of the point-to-point distances according to the local statistics of the neighborhoods surrounding points $i$ and $j$.

The selection of the local scale $\sigma_i$ can be done by studying the local statistics of the neighborhood of point $s_i$. A simple choice, which is used for the experiments in this paper, is:

$$\sigma_i = d(s_i, s_K) \tag{2}$$

where $s_K$ is the $K$'th neighbor of point $s_i$. The selection of $K$ is independent of scale and is a function of the data dimension of the embedding space. Nevertheless, in all our experiments (both on synthetic data and on images) we used a single value of $K = 7$, which gave good results even for high-dimensional data (the experiments with high-dimensional data were left out due to lack of space).

Figure 2 provides a visualization of the effect of the suggested local scaling. Since the data resides in multiple scales (one cluster is tight and the other is sparse) the standard approach to estimating affinities fails to capture the data structure (see Figure 2.b). Local scaling automatically finds the two scales and results in high affinities within clusters and low affinities across clusters (see Figure 2.c). This is the information required for separation.

We tested the power of local scaling by clustering the data set of Figure 1, plus four additional examples. We modified the Ng-Jordan-Weiss algorithm reviewed in Section 1.1 substituting the locally scaled affinity matrix $\hat{A}$ (of Eq. (1)) instead of $A$. Results are shown in Figure 3. In spite of the multiple scales and the various types of structure, the groups now match the intuitive solution.

## 3 Estimating the Number of Clusters

Having defined a scheme to set the scale parameter automatically we are left with one more free parameter: the number of clusters. This parameter is usually set manually and

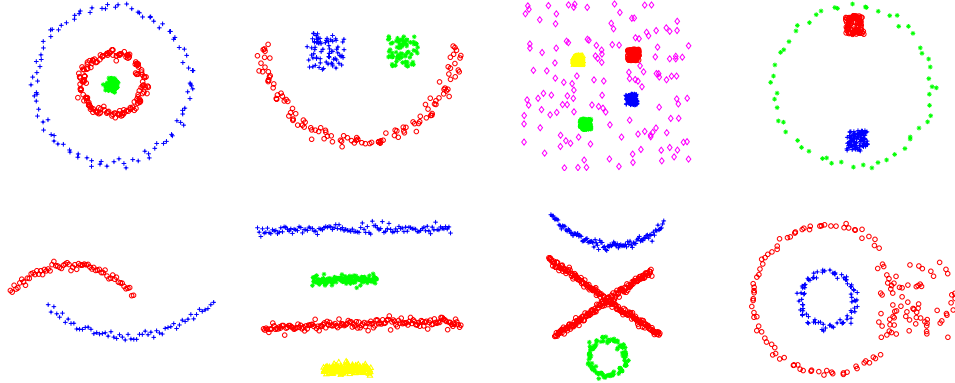

Figure 3: **Our clustering results.** *Using the algorithm summarized in Section 4. The number of groups was found automatically.*

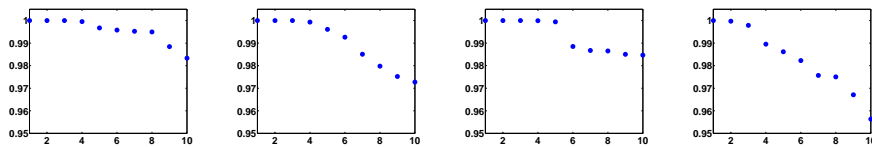

Figure 4: **Eigenvalues.** *The first 10 eigenvalues of $L$ corresponding to the top row data sets of Figure 3.*

not much research has been done as to how might one set it automatically. In this section we suggest an approach to discovering the number of clusters. The suggested scheme turns out to lead to a new spatial clustering algorithm.

### 3.1 The Intuitive Solution: Analyzing the Eigenvalues

One possible approach to try and discover the number of groups is to analyze the eigenvalues of the affinity matrix. The analysis given in [5] shows that the first (highest magnitude) eigenvalue of $L$ (see Section 1.1) will be a repeated eigenvalue of magnitude 1 with multiplicity equal to the number of groups $C$. This implies one could estimate $C$ by counting the number of eigenvalues equaling 1.

Examining the eigenvalues of our locally scaled matrix, corresponding to clean data-sets, indeed shows that the multiplicity of eigenvalue 1 equals the number of groups. However, if the groups are not clearly separated, once noise is introduced, the values start to deviate from 1, thus the criterion of choice becomes tricky. An alternative approach would be to search for a drop in the magnitude of the eigenvalues (this was pursued to some extent by Polito and Perona in [7]). This approach, however, lacks a theoretical justification. The eigenvalues of $L$ are the union of the eigenvalues of the sub-matrices corresponding to each cluster. This implies the eigenvalues depend on the structure of the individual clusters and thus no assumptions can be placed on their values. In particular, the gap between the $C$'th eigenvalue and the next one can be either small or large. Figure 4 shows the first 10 eigenvalues corresponding to the top row examples of Figure 3. It highlights the different patterns of distribution of eigenvalues for different data sets.

### 3.2 A Better Approach: Analyzing the Eigenvectors

We thus suggest an alternative approach which relies on the structure of the *eigenvectors*. After sorting $L$ according to clusters, in the "ideal" case (i.e., when $L$ is strictly block diagonal with blocks $L^{(c)}, c = 1, \ldots, C$), its eigenvalues and eigenvectors are the union of the eigenvalues and eigenvectors of its blocks padded appropriately with zeros (see [6, 5]). As long as the eigenvalues of the blocks are different each eigen-

vector will have non-zero values only in entries corresponding to a single block/cluster:

$$\hat{X} = \begin{bmatrix} x^{(1)} & \vec{0} & \vec{0} \\ \vec{0} & \cdots & \vec{0} \\ \vec{0} & \vec{0} & x^{(C)} \end{bmatrix}_{n \times C}$$ where $x^{(c)}$ is an eigenvector of the sub-matrix $L^{(c)}$ corresponding to cluster $c$. However, as was shown above, the eigenvalue 1 is bound to be a repeated eigenvalue with multiplicity equal to the number of groups $C$. Thus, the eigensolver could just as easily have picked any other set of orthogonal vectors spanning the same subspace as $\hat{X}$'s columns. That is, $\hat{X}$ could have been replaced by $X = \hat{X}R$ for any orthogonal matrix $R \in \mathcal{R}^{C \times C}$.

This, however, implies that even if the eigensolver provided us the rotated set of vectors, we are still guaranteed that there exists a rotation $\hat{R}$ such that each row in the matrix $X\hat{R}$ has a single non-zero entry. Since the eigenvectors of $L$ are the union of the eigenvectors of its individual blocks (padded with zeros), taking more than the first $C$ eigenvectors will result in more than one non-zero entry in some of the rows. Taking fewer eigenvectors we do not have a full basis spanning the subspace, thus depending on the initial $X$ there might or might not exist such a rotation. Note, that these observations are independent of the difference in magnitude between the eigenvalues.

We use these observations to predict the number of groups. For each possible group number $C$ we recover the rotation which best aligns $X$'s columns with the canonical coordinate system. Let $Z \in \mathcal{R}^{n \times C}$ be the matrix obtained after rotating the eigenvector matrix $X$, i.e., $Z = XR$ and denote $M_i = \max_j Z_{ij}$. We wish to recover the rotation $R$ for which in every row in $Z$ there will be at most one non-zero entry. We thus define a cost function:

$$J = \sum_{i=1}^{n} \sum_{j=1}^{C} \frac{Z_{ij}^2}{M_i^2} \tag{3}$$

Minimizing this cost function over all possible rotations will provide the best alignment with the canonical coordinate system. This is done using the gradient descent scheme described in Appendix A. The number of groups is taken as the one providing the minimal cost (if several group numbers yield practically the same minimal cost, the largest of those is selected).

The search over the group number can be performed incrementally saving computation time. We start by aligning the top two eigenvectors (as well as possible). Then, at each step of the search (up to the maximal group number), we add a single eigenvector to the already rotated ones. This can be viewed as taking the alignment result of the previous group number as an initialization to the current one. The alignment of this new set of eigenvectors is extremely fast (typically a few iterations) since the initialization is good. The overall run time of this incremental procedure is just slightly longer than aligning all the eigenvectors in a non-incremental way.

Using this scheme to estimate the number of groups on the data set of Figure 3 provided a correct result for all but one (for the right-most dataset at the bottom row we predicted 2 clusters instead of 3). Corresponding plots of the alignment quality for different group numbers are shown in Figure 5.

Yu and Shi [11] suggested rotating normalized eigenvectors to obtain an optimal segmentation. Their method iterates between non-maximum suppression (i.e., setting $M_i = 1$ and $Z_{ij} = 0$ otherwise) and using SVD to recover the rotation which best aligns the columns of $X$ with those of $Z$. In our experiments we noticed that this iterative method can easily get stuck in local minima and thus does not reliably find the optimal alignment and the group number. Another related approach is that suggested by Kannan et al. [3] who assigned points to clusters according to the maximal entry in the corresponding row of the eigenvector matrix. This works well when there are no repeated eigenvalues as then the eigenvectors

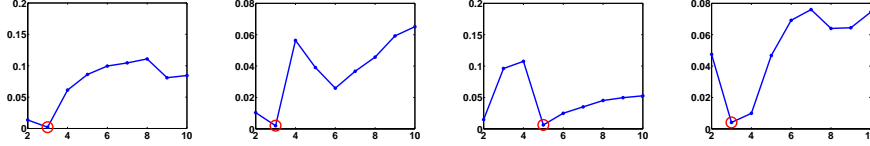

Figure 5: **Selecting Group Number.** *The alignment cost (of Eq. (3)) for varying group numbers corresponding to the top row data sets of Figure 3. The selected group number marked by a red circle, corresponds to the largest group number providing minimal cost (costs up to 0.01% apart were considered as same value).*

corresponding to different clusters are not intermixed. Kannan et al. used a non-normalized affinity matrix thus were not certain to obtain a repeated eigenvalue, however, this could easily happen and then the clustering would fail.

## 4   A New Algorithm

Our proposed method for estimating the number of groups automatically has two desirable by-products: (i) After aligning with the canonical coordinate system, one can use non-maximum suppression on the rows of $Z$, thus eliminating the final iterative k-means process, which often requires around 100 iterations and depends highly on its initialization. (ii) Since the final clustering can be conducted by non-maximum suppression, we obtain clustering results for all the inspected group numbers at a tiny additional cost. When the data is highly noisy, one can still employ k-means, or better, EM, to cluster the rows of $Z$. However, since the data is now aligned with the canonical coordinate scheme we can obtain by non-maximum suppression an excellent initialization so very few iterations suffice. We summarize our suggested algorithm:

**Algorithm:** Given a set of points $S = \{s_1, \ldots, s_n\}$ in $\mathcal{R}^l$ that we want to cluster:

1. Compute the local scale $\sigma_i$ for each point $s_i \in S$ using Eq. (2).

2. Form the locally scaled affinity matrix $\hat{A} \in \mathcal{R}^{n \times n}$ where $\hat{A}_{ij}$ is defined according to Eq. (1) for $i \neq j$ and $\hat{A}_{ii} = 0$.

3. Define $D$ to be a diagonal matrix with $D_{ii} = \sum_{j=1}^n \hat{A}_{ij}$ and construct the normalized affinity matrix $L = D^{-1/2} \hat{A} D^{-1/2}$.

4. Find $x_1, \ldots, x_C$ the $C$ largest eigenvectors of $L$ and form the matrix $X = [x_1, \ldots, x_C] \in \mathcal{R}^{n \times C}$, where $C$ is the largest possible group number.

5. Recover the rotation $R$ which best aligns $X$'s columns with the canonical coordinate system using the incremental gradient descent scheme (see also Appendix A).

6. Grade the cost of the alignment for each group number, up to $C$, according to Eq. (3).

7. Set the final group number $C_best$ to be the largest group number with minimal alignment cost.

8. Take the alignment result $Z$ of the top $C_best$ eigenvectors and assign the original point $s_i$ to cluster $c$ if and only if $max_j(Z_{ij}^2) = Z_{ic}^2$.

9. If highly noisy data, use the previous step result to initialize k-means, or EM, clustering on the rows of $Z$.

We tested the quality of this algorithm on real data. Figure 6 shows intensity based image segmentation results. The number of groups and the corresponding segmentation were obtained automatically. In this case same quality of results were obtained using non-scaled affinities, however, this required manual setting of both $\sigma$ (different values for different images) and the number of groups, whereas our result required no parameter settings.

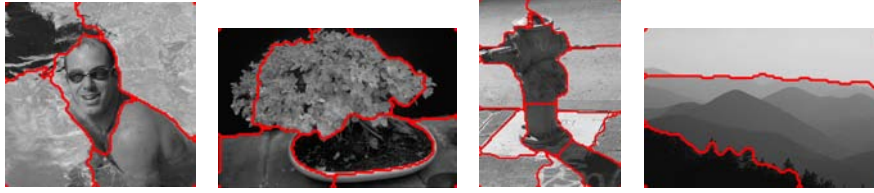

Figure 6: **Automatic image segmentation.** *Fully automatic intensity based image segmentation results using our algorithm.*

More experiments and results on real data sets can be found on our web-page **http://www.vision.caltech.edu/lihi/Demos/SelfTuningClustering.html**

## 5  Discussion & Conclusions

Spectral clustering practitioners know that selecting good parameters to tune the clustering process is an art requiring skill and patience. Automating spectral clustering was the main motivation for this study. The key ideas we introduced are three: (a) using a local scale, rather than a global one, (b) estimating the scale from the data, and (c) rotating the eigenvectors to create the maximally sparse representation. We proposed an automated spectral clustering algorithm based on these ideas: it computes automatically the scale and the number of groups and it can handle multi-scale data which are problematic for previous approaches.

Some of the choices we made in our implementation were motivated by simplicity and are perfectible. For instance, the local scale $\sigma$ might be better estimated by a method which relies on more informative local statistics. Another example: the cost function in Eq. (3) is reasonable, but by no means the only possibility (e.g. the sum of the entropy of the rows $Z_i$ might be used instead).

**Acknowledgments:**

Finally, we wish to thank Yair Weiss for providing us his code for spectral clustering. This research was supported by the MURI award number SA3318 and by the Center of Neuromorphic Systems Engineering award number EEC-9402726.

## References

[1] G. H. Golub and C. F. Van Loan "Matrix Computation", John Hopkins University Press, 1991, Second Edition.

[2] V. K. Goyal and M. Vetterli "Block Transform by Stochastic Gradient Descent" *IEEE Digital Signal Processing Workshop*, 1999, Bryce Canyon, UT, Aug. 1998

[3] R. Kannan, S. Vempala and V.Vetta "On Spectral Clustering – Good, Bad and Spectral" In *Proceedings of the 41st Annual Symposium on Foundations of Computer Sceince*, 2000.

[4] M. Meila and J. Shi "Learning Segmentation by Random Walks" *In Advances in Neural Information Processing Systems 13*, 2001

[5] A. Ng, M. Jordan and Y. Weiss "On spectral clustering: Analysis and an algorithm" *In Advances in Neural Information Processing Systems 14*, 2001

[6] P. Perona and W. T. Freeman "A Factorization Approach to Grouping" *Proceedings of the 5th European Conference on Computer Vision*, Volume I, pp. 655–670 1998.

[7] M. Polito and P. Perona "Grouping and dimensionality reduction by locally linear embedding" *Advances in Neural Information Processing Systems 14*, 2002

[8] G.L. Scott and H.C. Longuet-Higgins "Feature grouping by 'relocalisation' of eigenvectors of the proximity matrix" *In Proc. British Machine Vision Conference*, Oxford, UK, pages 103–108, 1990.

[9] J. Shi and J. Malik "Normalized Cuts and Image Segmentation" *IEEE Transactions on Pattern Analysis and Machine Intelligence*, 22(8), 888-905, August 2000.

[10] Y. Weiss "Segmentation Using Eigenvectors: A Unifying View" *International Conference on Computer Vision*, pp.975–982,September,1999.

[11] S. X. Yu and J. Shi "Multiclass Spectral Clustering" *International Conference on Computer Vision*, Nice, France, pp.11–17,October,2003.

## A  Recovering the Aligning Rotation

To find the best alignment for a set of eigenvectors we adopt a gradient descent scheme similar to that suggested in [2]. There, Givens rotations where used to recover a rotation which diagonalizes a symmetric matrix by minimizing a cost function which measures the diagonality of the matrix. Similarly, here, we define a cost function which measures the alignment quality of a set of vectors and prove that the gradient descent, using Givens rotations, converges.

The cost function we wish to minimize is that of Eq. (3). Let $m_i = j$ such that $Z_{ij} = Z_{im_i} = M_i$. Note, that the indices $m_i$ of the maximal entries of the rows of $X$ might be different than those of the optimal $Z$. A simple non-maximum supression on the rows of $X$ can provide a wrong result. Using the gradient descent scheme allows to increase the cost corresponding to part of the rows as long as the overall cost is reduced, thus enabling changing the indices $m_i$.

Similar to [2] we wish to represent the rotation matrix $R$ in terms of the smallest possible number of parameters. Let $\tilde{G}_{i,j,\theta}$ denote a Givens rotation [1] of $\theta$ radians (counterclockwise) in the $(i,j)$ coordinate plane. It is sufficient to consider Givens rotations so that $i < j$, thus we can use a convenient index re-mapping $G_{k,\theta} = \tilde{G}_{i,j,\theta}$, where $(i,j)$ is the $k$th entry of a lexicographical list of $(i,j) \in \{1,2,\dots,C\}^2$ pairs with $i < j$. Hence, finding the aligning rotation amounts to minimizing the cost function $J$ over $\Theta \in [-\pi/2, \pi/2)^K$. The update rule for $\Theta$ is: $\Theta_{k+1} = \Theta_k - \alpha \left. \nabla J \right|_{\Theta=\Theta_k}$ where $\alpha \in \mathcal{R}^+$ is the step size.

We next compute the gradient of $J$ and bounds on $\alpha$ for stability. For convenience we will further adopt the notation convention of [2]. Let $U_{(a,b)} = G_{a,\theta_a} G_{a+1,\theta_{a+1}} \cdots G_{b,\theta_b}$ where $U_{(a,b)} = I$ if $b < a$, $U_k = U_{(k,k)}$, and $V_k = \frac{\partial}{\partial \theta_k} U_k$. Define $A^{(k)}$, $1 \le k \le K$, element wise by $A^{(k)}_{ij} = \frac{\partial Z_{ij}}{\partial \theta_k}$. Since $Z = XR$ we obtain $A^{(k)} = X U_{(1,k-1)} V_k U_{(k+1,K)}$.

We can now compute $\nabla J$ element wise:
$$\frac{\partial J}{\partial \theta_k} = \sum_{i=1}^{n} \sum_{j=1}^{C} \frac{\partial}{\partial \theta_k} \frac{Z_{ij}^2}{M_i^2} - 1 = 2 \sum_{i=1}^{n} \sum_{j=1}^{C} \frac{Z_{ij}}{M_i^2} A^{(k)}_{ij} - \frac{Z_{ij}^2}{M_i^3} \frac{\partial M_i}{\partial \theta_k}$$

Due to lack of space we cannot describe in full detail the complete convergence proof. We thus refer the reader to [2] where it is shown that convergence is obtained when $1 - \alpha F_{kl}$ lie in the unit circle, where $F_{kl} = \left[ \frac{\partial^2 J}{\partial \theta_l \partial \theta_k} \right]_{\Theta=0}$. Note, that at $\Theta = 0$ we have $Z_{ij} = 0$ for $j \ne m_i$, $Z_{im_i} = M_i$, and $\frac{\partial M_i}{\partial \theta_k} = \frac{\partial Z_{im_i}}{\partial \theta_k} = A^{(k)}_{im_i}$ (i.e., near $\Theta = 0$ the maximal entry for each row does not change its index). Deriving thus gives $\left[ \frac{\partial^2 J}{\partial \theta_l \partial \theta_k} \right]_{ij|\Theta=0} = 2 \sum_{i=1}^{n} \sum_{j \ne m_i} \frac{1}{M_i^2} A^{(k)}_{ij} A^{(l)}_{ij}$. Further substituting in the values for $A^{(k)}_{ij}|_{\Theta=0}$ yields:

$$F_{kl} = \left[ \frac{\partial^2 J}{\partial \theta_l \partial \theta_k} \right]_{ij|\Theta=0} = \begin{cases} 2\#i \text{ s.t. } m_i = i_k \text{ or } m_i = j_k & \text{if } k = l \\ 0 & \text{otherwise} \end{cases}$$

where $(i_k, j_k)$ is the pair $(i,j)$ corresponding to the index $k$ in the index re-mapping discussed above. Hence, by setting $\alpha$ small enough we get that $1 - \alpha F_{kl}$ lie in the unit circle and convergence is guaranteed.